# An Oculo-Motor System with Multi-Chip Neuromorphic Analog VLSI Control

**Oliver Landolt***
CSEM SA
2007 Neuchâtel / Switzerland
E-mail: landolt@caltech.edu

**Stève Gyger**
CSEM SA
2007 Neuchâtel / Switzerland
E-mail: steve.gyger@csem.ch

## Abstract

A system emulating the functionality of a moving eye—hence the name *oculo-motor system*—has been built and successfully tested. It is made of an optical device for shifting the field of view of an image sensor by up to 45° in any direction, four neuromorphic analog VLSI circuits implementing an oculo-motor control loop, and some off-the-shelf electronics. The custom integrated circuits communicate with each other primarily by non-arbitrated address-event buses. The system implements the behaviors of saliency-based *saccadic exploration*, and *smooth pursuit* of light spots. The duration of saccades ranges from 45 ms to 100 ms, which is comparable to human eye performance. Smooth pursuit operates on light sources moving at up to 50°/s in the visual field.

## 1 INTRODUCTION

Inspiration from biology has been recognized as a seminal approach to address some engineering challenges, particularly in the computational domain [1]. Researchers have borrowed architectures, operating principles and even micro-circuits from various biological neural structures and turned them into analog VLSI circuits [2]. Neuromorphic approaches are often considered to be particularly suited for machine vision, because even simple animals are fitted with neural systems that can easily outperform most sequential digital computers in visual processing tasks. It has long been recognized that the level of visual processing capability needed for practical applications would require more circuit area than can be fitted on a single chip. This observation has triggered the development of inter-chip communication schemes suitable for neuromorphic analog VLSI circuits [3]-[4], enabling the combination of several chips into a system capable of addressing tasks of higher complexity. Despite the availability of these communication protocols, only few successful implementations of multi-chip neuromorphic systems have been reported so far (see [5] for a review). The present contribution reports the completion of a fully functional multi-chip system emulating the functionality of a moving eye, hence the denomination *oculo-motor system*. It is made of two 2D VLSI retina chips, two custom analog VLSI control chips, dedicated optical and mechanical devices and off-the-shelf electronic components. The four neuromorphic chips communicate mostly by pulse streams mediated by non-arbitrated address-event buses [4]. In its current version, the system can generate *saccades* (quick eye

movements) toward salient points of the visual scene, and track moving light spots. The purpose of the saccadic operating mode is to explore the visual scene efficiently by allocating processing time proportionally to significance. The purpose of tracking (also called *smooth pursuit*) is to slow down or suppress the retina image slip of moving objects in order to leave visual circuitry more time for processing. The two modes—saccadic exploration and smooth pursuit—operate concurrently and interact with each other. The development of this oculo-motor system was meant as a framework in which some general issues pertinent to neuromorphic engineering could be addressed. In this respect, it complements Horiuchi's pioneering work [6]-[7], which consisted of developing a 1D model of the primate oculo-motor system with a focus on automatic on-chip learning of the correct control function. The new system addresses different issues, notably 2D operation and the problem of strongly non-linear mapping between 2D visual and motor spaces.

## 2 SYSTEM DESCRIPTION

The oculo-motor system is made of three modules (Fig. 1). The moving eye module contains a 35 by 35 pixels electronic retina [8] fitted with a light deflection device driven by two motors. This device can shift the field of view of the retina by up to 45° in any direction. The optics are designed to cover only a narrow field of view of about 12°. Thereby, the retina serves as a high-resolution "spotlight" gathering details of interesting areas of the visual scene, similarly to the fovea of animals. Two position control loops implemented by off-the-shelf components keep the optical elements in the position specified by input signals applied to this module. The other modules control the moving eye in two types of behavior, namely *saccadic exploration* and *smooth pursuit*. They are implemented as physically distinct printed circuit boards which can be enabled or disabled independently.

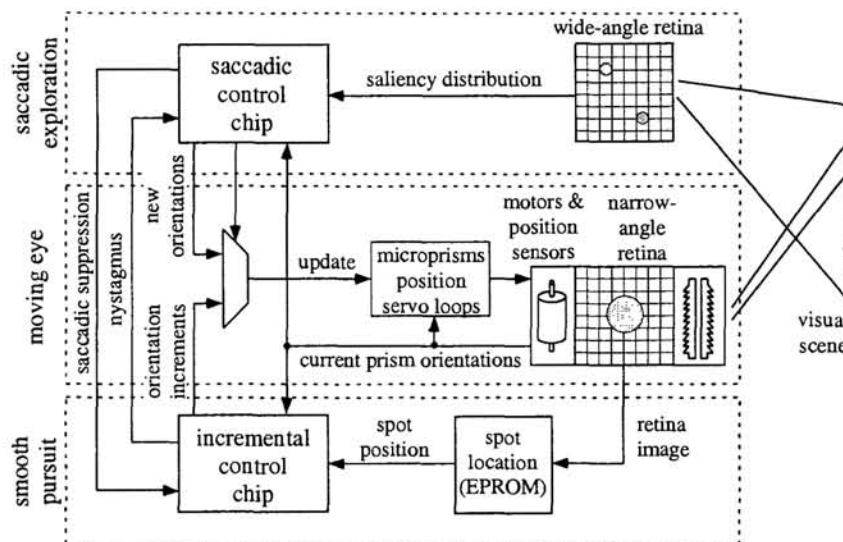

Figure 1: Oculo-motor system architecture

The light deflection device is made of two transparent and flat disks with a micro-prism grating on one side, mounted perpendicularly to the optical axis of a lens. Each disk can rotate without restriction around this axis, independently from the other. As a whole, each micro-prism grating acts on light essentially like a single large prism, except that it takes much less space (Fig. 2). Although a single fixed prism cannot have an adjustable deflection angle, with two mobile prisms, any magnitude and direction of deflection within some boundary can be selected, because the two contributions may combine either con-

structively or destructively depending on the relative prism orientations. The relationship between prism orientations and deflection angle has been derived in [9]. The advantage of this system over many other designs is that only two small passive optical elements have to move whereas most of the components are fixed, which enables fast movements and avoids electrical connections to moving parts. The drawback of this principle is that optical aberrations introduced by the prisms degrade image quality. However, when the device is used in conjunction with a typical electronic retina, this degradation is not limiting because these image sensors are characterized by a modest resolution due to focal-plane electronic processing.

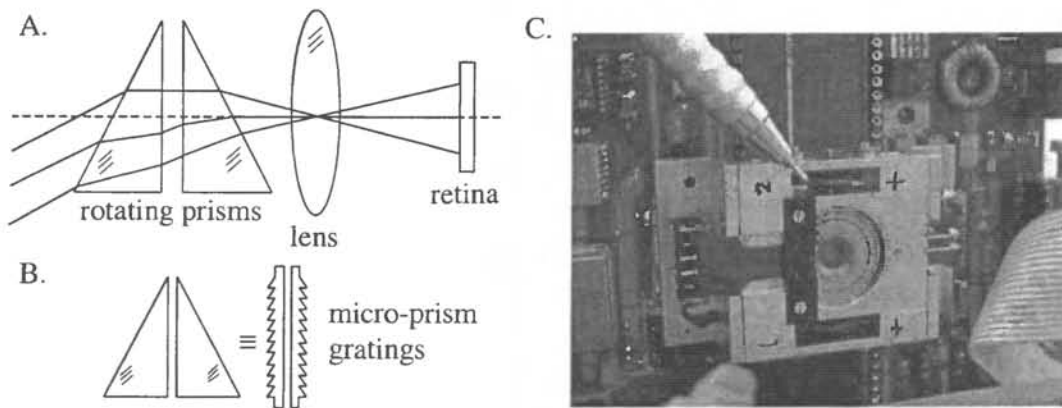

Figure 2: A. Light deflection device principle. B. Replacement of conventional prisms by micro-prism gratings. C. Photograph of the prototype with motors and orientation sensors.

The saccadic exploration module (Fig. 1) consists of an additional retina fitted with a fixed wide-angle lens, and a neuromorphic saccadic control chip. The retina gathers low-resolution information from the whole visual scene accessible to the moving eye, determines the degree of interest—or *saliency* [10]—of every region and transmits the resulting saliency distribution to the saccadic control chip. In the current version of the system, the distribution of saliency is just the raw output image of the retina, whereby saliency is determined by the brightness of visual scene locations. By inserting additional visual processing hardware between the retina and the saccadic control chip, it would be possible to generate interest for more sophisticated cues like edges, motion or specific shapes or patterns. The saccadic control chip (Fig. 3) determines the sequence and timing of an endless succession of quick jumps—or *saccades*—to be executed by the moving eye, in such a way that salient locations are attended longer and more frequently than less significant locations. The chip contains a 2D array of about 900 cells, which is called *visual map* because its organization matches the topology of the visual field accessible by the moving eye. The chip also contains two 1D arrays of 64 cells called *motor maps*, which encode micro-prism orientations in the light deflection device. Each cell of the visual map is externally stimulated by a stream of brief pulses, the frequency of which encodes saliency. The cells integrate incoming pulses over time on a capacitor, thereby building up an internal voltage at a rate proportional to pulse frequency. A global comparison circuit—called *winner-take-all*—selects the cell with the highest internal voltage. In the winning cell, a leakage mechanism slowly decrease the internal voltage over time, thereby eventually leading another cell to win. With this principle, any cell stimulated to some degree wins from time to time. The frequency of winning and the time ellapsed until another cell wins increases with saliency. The visual map and the two motor maps are interconnected by a so-called *network of links* [9], which embodies the mapping between visual and motor spaces. This network consists of a pair of wires running from each visual cell to one cell in each of the two motor maps. Thereby, the winning cell in the visual map stimulates exactly one cell in

each motor map. The *location* of the active cell in a motor map encodes the orientation of a micro-prism grating, therefore this representation convention is called *place coding* [9]. The addresses of the active cells on the motor maps are transmitted to the moving eye, which triggers micro-prism displacements toward the specified orientations.

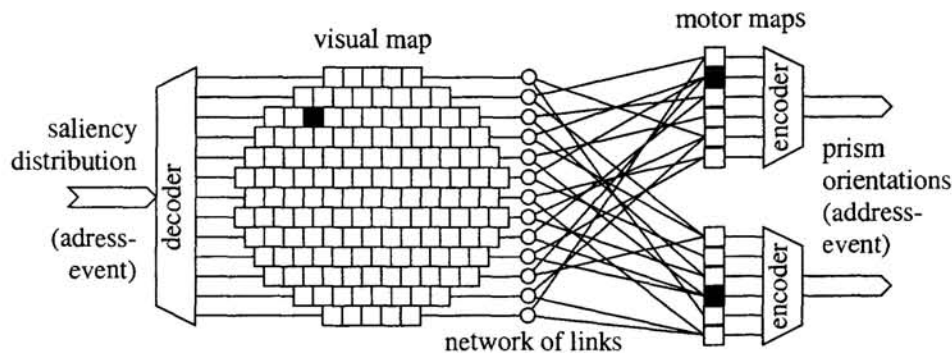

Figure 3: Schematic of the saccadic control chip

The smooth pursuit module consists of an EPROM chip and a neuromorphic incremental control chip (Fig. 1). The address-event stream delivered by the narrow-field retina is applied to the EPROM. The field of view of this retina has been divided up into eight angular sectors and a center region (Fig. 4A). The EPROM maps the addresses of pixels located in the same sector onto a common output address, thereby summing their spiking frequencies. The resulting address-event stream is applied to a topological map of eight cells constituting one of the inputs of the neuromorphic incremental control chip. If a single bright spot is focused on the retina away from the center, a large sum is produced in one or two neighboring cells of this map, whereas the other cells receive only background stimulation levels close to zero. Thereby, the angular position of the light spot is encoded by the location of the spot of activity on the map—in other words place coding. Other objects than light spots could be processed similarly after insertion of relevant detection hardware between the retina and the EPROM. The incremental control chip has two additional input maps representing the current orientations of the two prisms (Fig. 4B). These maps are connected to position sensors incorporated into the moving eye module (Fig. 1). These additional inputs are necessary because the control actions depends not only on the location of the target on the retina, but also on the current prism orientations [9]. The control actions are computed by three networks of links relating the primary inputs maps to the final output map via an intermediate layer. The purpose of this intermediate stage is to break down the control function of three variables into three functions of only two variables, which can be implemented by a lower number of links [11]. As in the saccadic control chip, the mapping between the input and output spaces has been calculated numerically prior to chip fabrication, then hardwired as electrical connections. The final outputs of the chip are pulse streams encoding the direction and rate at which each micro-prism grating must rotate in order to shift the target toward the center of the retina. These pulses incrementally update prism orientations settings at the input of the moving eye module (Fig. 1).

Since two different modules control the same moving eye, it is necessary to coordinate them in order to avoid conflicts. Saccadic module interventions occur whenever a saccade is generated, namely every 200–500 ms in typical operating conditions. At the instant a saccade is requested, the smooth pursuit module is shut off in order to prevent it from reacting against the saccade. A similar mechanism called *saccadic suppression* exists in biology. When the eye reaches the target location, control is left entirely to the smooth pursuit module until the next saccade is generated. Reciprocally, if an object tracked by

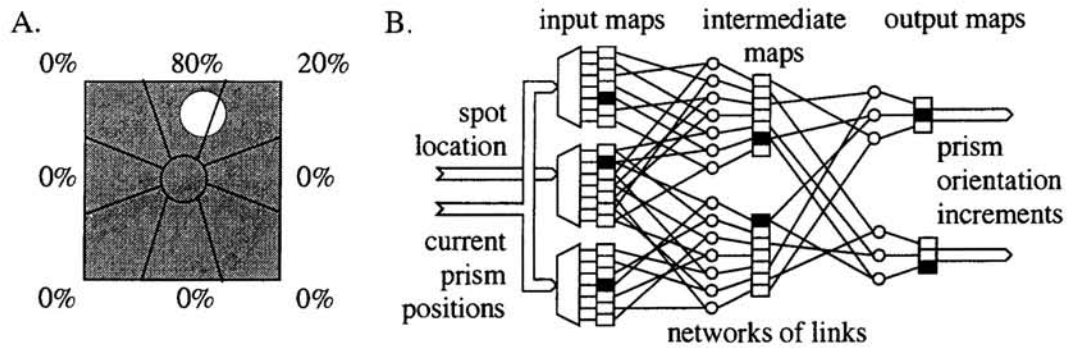

Figure 4: A. Place-coded spot location obtained by summing the outputs of pixels belonging to the same sector. B. Architecture of the incremental control chip

the smooth pursuit module reaches the boundary of the global visual field, the incremental control chip sends a signal triggering a saccade back toward the center of the visual field—which is called *nystagmus* in biology. The reason for splitting control into two modules is that visuo-motor coordinate mappings are very different for saccadic exploration and for smooth pursuit [9]. In the former case, visual input is related to the global field of view covered by the fixed wide-angle retina, and outputs are absolute micro-prism orientations. Saccade targets need not be initially visible to the moving eye. Since saccades are executed without permanent visual feedback, their accuracy is limited by the mapping hardwired in the control chip. Inversely, smooth pursuit is based on information extracted directly from the retina image of the moving eye. The output of the incremental control chip are small changes in micro-prism orientations instead of absolute positions. Thereby, the smooth pursuit module operates under closed-loop visual feedback, which confers it high accuracy. However, operation under visual feedback is slower than open-loop saccadic movements, and smooth pursuit inherently applies only to a single target. Thus, the two control modules are very complementary in purpose and performance.

## 3   EXPERIMENTAL RESULTS

The present section reports both qualitative observations and quantitative measurements made on the oculo-motor system, because the complexity of its behavior is difficult to convey by just a few numbers. The measurement setup consisted of a black board on which high efficiency white light emitting diodes were mounted, the intensity of which could be set individually. The visual scene was placed about 70 cm away from the moving eye. The axes of the two retinas were parallel at a distance of 6.5 cm. It was necessary to take this spacing into account for the visuo-motor coordinate mapping. The saliency distribution produced by the visual scene was measured by analyzing the output image of the wide-angle retina chip (Fig. 1).

When a single torchlight was waved in front of the moving eye, it was found that the smooth pursuit system indeed keeps the center of gravity of the light source image at the center of the narrow field of view. The maximum tracking velocity depends on the intensity ratio—contrast—between the light spot and the background. This behavior was expected because by construction, the incremental control chip generates correction pulses at a rate proportional to the magnitude of its input signals. At the highest contrast, we were able to achieve a maximum tracking speed of 50 °/s. For comparison, smooth pursuit in humans can in principle reach up to 180 °/s, but tracking is accurate only up to about 30 °/s [7].

When shown two fixed light spots, the moving eye jumps from one to the other periodically.

The relative time spent on each light source depends on their intensity ratio. The duty cycle has been measured for ratios ranging from 0.1 to 10 (Fig. 5A). It is close to 50% for equal saliency, and tends toward a ratio of 10 to 1 in favor of the brightest spot at the extremities of the range. The delay between onset of a saccade and stabilization on the target ranges from 45 ms to 100 ms. The delay is not constant because it depends to some extent on saccade magnitude, and because of occasional mechanical slipping at the onset. In humans, the duration of saccades tends to be proportional to their amplitude, and ranges between 25 ms and 200 ms.

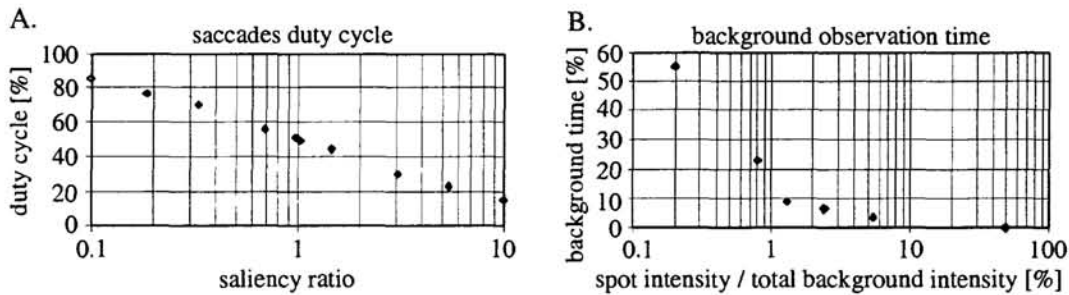

Figure 5: Measured data plots. A. Gaze time sharing between two salient spots versus saliency ratio. B. Gaze time on background versus spot-to-background intensity ratio.

When more than two spots are turned on, the saccadic exploration is not obviously periodic anymore, but the eye keeps spending most time on the light spots, with a noticeable preference for larger intensities. This behavior is consistent with measurements previously made on the saccadic control chip alone under electrical stimulation [9]. Saccades towards locations in the background are rare and brief if the intensity ratio between the light sources and the background is high enough. This phenomenon has been studied quantitatively by measuring the fraction of time spent on background locations for different light source intensities (Fig. 5B). The quantity on the horizontal axis of the plot is the ratio between the total intensity in light spots and the total background intensity. These two quantities are measured by summing the outputs of wide-angle retina pixels belonging to the light spot images and to the background respectively. It can be seen that if this ratio is above 1, less than 10% of the time is spent scanning the background.

Open-loop saccade accuracy has been evaluated by switching off the smooth pursuit module, and measuring the error vector between the center of gravity of the light spot and the center of the narrow-field retina after each saccade, for six different light spots spread over the field of view. The error vectors were found to be always less than 2° in magnitude, with different orientations in each case. Whenever the moving eye returned to a same light spot, the error vector was the same. This shows that the residual error is not due to random noise, but to the limited accuracy of visuo-motor mapping within the saccadic control chip. The magnitude of the error is always low enough that the target light spot is completely visible by the moving eye, thereby ensuring that the smooth pursuit module can indeed correct the error when enabled.

## 4   CONCLUSION

The oculo-motor system described herein performs as intended, thereby demonstrating the value of a neuromorphic engineering approach in the case of a relatively complex task involving mechanical and optical components. This system provides an experimental platform for studying *active vision*, whereby a visual system acts on itself in order to facilitate perception of its surroundings. Besides saccadic exploration and smooth pursuit, a mov-

ing eye can be exploited to improve vision in many other ways. For instance, resolution shortcomings in retinas incorporating only a modest number of pixels can be overcome by continuously sweeping the field of view back and forth, thereby providing continuous information in space—although not simultaneously in time. In binocular vision, 3D information perception by stereopsis is also made easier if the fields of view can be aligned by vergence control [12]. Besides active vision, the oculo-motor system also lends itself as a framework for testing and demonstrating other analog VLSI vision circuits. As already mentioned, due to its modular architecture, it is possible to insert additional visual processing chips either in the saccadic exploration module, or in the smooth pursuit module, in order to make the current light-source oriented system suitable for operation in natural visual environments.

## Acknowledgments

The authors wish to express their gratitude to all their colleagues at CSEM who contributed to this work. Special thanks are due to Patrick Debergh for the micro-prism light deflection concept, to Friedrich Heitger for designing and building the mechanical device, and to Edoardo Franzi for designing and building the related electronic interface. Thanks are also due to Arnaud Tisserand, Friedrich Heitger, Eric Vittoz, Reid Harrison, Theron Stanford, and Edoardo Franzi for helpful comments on the manuscript. Mr. Roland Lagger, from Portescap, La Chaux-de-Fonds, Switzerland, provided friendly assistance in a critical mechanical assembly step.

## Footnotes

* Now with Koch Lab, Division of Biology 139-74, Caltech, Pasadena, CA 91125, USA

## References

[1] C. Mead. *Analog VLSI and Neural Systems*. Addison Wesley, 1989.

[2] T.S. Lande, editor. *Neuromorphic Systems Engineering*. Kluwer Academic Publishers, Dordrecht, 1998.

[3] K. Boahen. Retinomorphic vision systems II: Communication channel design. In *IEEE Int. Symp. Circuits and Systems (ISCAS'96)*, Atlanta, May 1996.

[4] A. Mortara, E. Vittoz, and P. Venier. A communication scheme for analog VLSI perceptive systems. *IEEE Journal of Solid-State Circuits*, 30, June 1995.

[5] C.M. Higgins. Multi-chip neuromorphic motion processing. In *Conference on Advanced Research in VLSI*, Atlanta, March 1999.

[6] T.K. Horiuchi, B. Bishofberger, and C. Koch. An analog VLSI saccadic eye movement system. In *Advances in Neural Processing Systems 6*, 1994.

[7] T.K. Horiuchi. *Analog VLSI-Based, Neuromorphic Sensorimotor Systems: Modeling the Primate Oculomotor System*. PhD thesis, Caltech, Pasadena, 1997.

[8] P. Venier. A constrast sensitive silicon retina based on conductance modulation in a diffusion network. In *6th Int. Conf. Microelectronics for Neural Networks and Fuzzy Systems (MicroNeuro'97)*, Dresden, Sept 1997.

[9] O. Landolt. *Place Coding in Analog VLSI - A Neuromorphic Approach to Computation*. Kluwer Academic Publishers, Dordrecht, 1998.

[10] T.G. Morris and S.P. DeWeerth. Analog VLSI excitatory feedback circuits for attentional shifts and tracking. *Analog Integrated Circuits and Signal Processing*, 13, May-June 1997.

[11] O. Landolt. Place coding in analog VLSI and its application to the control of a light deflection system. In *MicroNeuro'97*, Dresden, Sept 1997.

[12] M. Mahowald. *An Analog VLSI System for Stereoscopic Vision*. Kluwer Academic Publishers, Boston, 1994.